# On the Design of Loss Functions for Classification: theory, robustness to outliers, and SavageBoost

**Hamed Masnadi-Shirazi**
Statistical Visual Computing Laboratory,
University of California, San Diego
La Jolla, CA 92039
hmasnadi@ucsd.edu

**Nuno Vasconcelos**
Statistical Visual Computing Laboratory,
University of California, San Diego
La Jolla, CA 92039
nuno@ucsd.edu

## Abstract

The machine learning problem of classifier design is studied from the perspective of probability elicitation, in statistics. This shows that the standard approach of proceeding from the specification of a loss, to the minimization of conditional risk is overly restrictive. It is shown that a better alternative is to start from the specification of a functional form for the minimum conditional risk, and derive the loss function. This has various consequences of practical interest, such as showing that 1) the widely adopted practice of relying on convex loss functions is unnecessary, and 2) many new losses can be derived for classification problems. These points are illustrated by the derivation of a new loss which is not convex, but does not compromise the computational tractability of classifier design, and is robust to the contamination of data with outliers. A new boosting algorithm, SavageBoost, is derived for the minimization of this loss. Experimental results show that it is indeed less sensitive to outliers than conventional methods, such as Ada, Real, or LogitBoost, and converges in fewer iterations.

## 1 Introduction

The binary classification of examples $\mathbf{x}$ is usually performed with recourse to the mapping $\hat{y} = sign[f(\mathbf{x})]$, where $f$ is a function from a pre-defined class $\mathcal{F}$, and $\hat{y}$ the predicted class label. Most state-of-the-art classifier design algorithms, including SVMs, boosting, and logistic regression, determine the optimal function $f^*$ by a three step procedure: 1) define a loss function $\phi(yf(\mathbf{x}))$, where $y$ is the class label of $\mathbf{x}$, 2) select a function class $\mathcal{F}$, and 3) search within $\mathcal{F}$ for the function $f^*$ which minimizes the expected value of the loss, known as minimum conditional risk. Although tremendously successful, these methods have been known to suffer from some limitations, such as slow convergence, or too much sensitivity to the presence of outliers in the data [1, 2]. Such limitations can be attributed to the loss functions $\phi(\cdot)$ on which the algorithms are based. These are convex bounds on the so-called *0-1 loss*, which produces classifiers of minimum probability of error, but is too difficult to handle from a computational point of view.

In this work, we analyze the problem of classifier design from a different perspective, that has long been used to study the problem of probability elicitation, in the statistics literature. We show that the two problems are identical, and probability elicitation can be seen as a reverse procedure for solving the classification problem: 1) define the functional form of expected elicitation loss, 2) select a function class $\mathcal{F}$, and 3) derive a loss function $\phi$. Both probability elicitation and classifier design reduce to the problem of minimizing a Bregman divergence. We derive equivalence results, which allow the representation of the classifier design procedures in "probability elicitation form", and the representation of the probability elicitation procedures in "machine learning form". This equivalence is useful in two ways. From the elicitation point of view, the risk functions used in machine learning can be used as new elicitation losses. From the machine learning point of view, new insights on the relationship between loss $\phi$, optimal function $f^*$, and minimum risk are obtained. In particular, it is shown that the classical progression from loss to risk is overly restrictive: once a loss $\phi$ is specified,

both the optimal $f^*$, and the functional form of the minimum risk are immediately pined down. This is, however, not the case for the reverse progression: it is shown that any functional form of the minimum conditional risk, which satisfies some mild constraints, supports many $(\phi, f^*)$ pairs. Hence, once the risk is selected, one degree of freedom remains: by selecting a class of $f^*$, it is possible to tailor the loss $\phi$, so as to guarantee classifiers with desirable traits. In addition to this, the elicitation view reveals that the machine learning emphasis on convex losses $\phi$ is misguided. In particular, it is shown that what matters is the convexity of the minimum conditional risk. Once a functional form is selected for this quantity, the convexity of the loss $\phi$ does not affect the convexity of the Bregman divergence to be optimized.

These results suggest that many new loss functions can be derived for classifier design. We illustrate this, by deriving a new loss that trades convexity for boundedness. Unlike all previous $\phi$, the one now proposed remains constant for strongly negative values of its argument. This is akin to robust loss functions proposed in the statistics literature to reduce the impact of outliers. We derive a new boosting algorithm, denoted SavageBoost, by combination of the new loss and the procedure used by Friedman to derive RealBoost [3]. Experimental results show that the new boosting algorithm is indeed more outlier resistant than classical methods, such as AdaBoost, RealBoost, and LogitBoost.

## 2 Classification and risk minimization

A classifier is a mapping $g : \mathcal{X} \rightarrow \{-1, 1\}$ that assigns a class label $y \in \{-1, 1\}$ to a feature vector $\mathbf{x} \in \mathcal{X}$, where $\mathcal{X}$ is some feature space. If feature vectors are drawn with probability density $P_{\mathbf{X}}(\mathbf{x})$, $P_Y(y)$ is the probability distribution of the labels $y \in \{-1, 1\}$, and $L(\mathbf{x}, y)$ a loss function, the classification risk is $R(f) = E_{\mathbf{X}, Y}[L(g(\mathbf{x}), y)]$. Under the *0-1 loss*, $L_{0/1}(\mathbf{x}, y) = 1$ if $g(\mathbf{x}) \neq y$ and 0 otherwise, this risk is the expected probability of classification error, and is well known to be minimized by the Bayes decision rule. Denoting by $\eta(\mathbf{x}) = P_{Y|\mathbf{X}}(1|\mathbf{x})$ this can be written as

$$g^*(\mathbf{x}) = sign[2\eta(\mathbf{x}) - 1]. \tag{1}$$

Classifiers are usually implemented with mappings of the form $g(\mathbf{x}) = sign[f(\mathbf{x})]$, where $f$ is some mapping from $\mathcal{X}$ to $\mathbb{R}$. The minimization of the *0-1 loss* requires that

$$sign[f^*(\mathbf{x})] = sign[2\eta(\mathbf{x}) - 1], \ \ \forall \mathbf{x} \tag{2}$$

When the classes are separable, any $f(\mathbf{x})$ such that $yf(\mathbf{x}) \geq 0, \forall \mathbf{x}$ has zero classification error. The *0-1 loss* can be written as a function of this quantity

$$L_{0/1}(\mathbf{x}, y) = \phi_{0/1}[yf(\mathbf{x})] = sign[-yf(\mathbf{x})].$$

This motivates the minimization of the expected value of this loss as a goal for machine learning. However, this minimization is usually difficult. Many algorithms have been proposed to minimize alternative risks, based on convex upper-bounds of the *0-1 loss*. These risks are of the form

$$R_\phi(f) = E_{\mathbf{X}, Y}[\phi(yf(\mathbf{x}))] \tag{3}$$

where $\phi(\cdot)$ is a convex upper bound of $\phi_{0/1}(\cdot)$. Some examples of $\phi(\cdot)$ functions in the literature are given in Table 1. Since these functions are non-negative, the risk is minimized by minimizing the conditional risk $E_{Y|\mathbf{X}}[\phi(yf(\mathbf{x}))|\mathbf{X} = \mathbf{x}]$ for every $\mathbf{x} \in \mathcal{X}$. This conditional risk can be written as

$$C_\phi(\eta, f) = \eta\phi(f) + (1 - \eta)\phi(-f), \tag{4}$$

where we have omitted the dependence of $\eta$ and $f$ on $\mathbf{x}$ for notational convenience.

Various authors have shown that, for the $\phi(\cdot)$ of Table 1, the function $f_\phi^*$ which minimizes (4)

$$f_\phi^*(\eta) = \arg\min_f C_\phi(\eta, f) \tag{5}$$

satisfies (2) [3, 4, 5]. These functions are also presented in Table 1. It can, in fact, be shown that (2) holds for any $f_\phi^*(\cdot)$ which minimizes (4) whenever $\phi(\cdot)$ is convex, differentiable at the origin, and has derivative $\phi'(0) = 0$ [5].

While learning algorithms based on the minimization of (4), such as SVMs, boosting, or logistic regression, can perform quite well, they are known to be overly sensitive to outliers [1, 2]. These are points for which $yf(\mathbf{x}) < 0$. As can be seen from Figure 1, the sensitivity stems from the large

Table 1: Machine learning algorithms progress from loss $\phi$, to inverse link function $f_\phi^*(\eta)$, and minimum conditional risk $C_\phi^*(\eta)$.

| Algorithm | $\phi(v)$ | $f_\phi^*(\eta)$ | $C_\phi^*(\eta)$ |
|---|---|---|---|
| Least squares | $(1-v)^2$ | $2\eta - 1$ | $4\eta(1-\eta)$ |
| Modified LS | $\max(1-v,0)^2$ | $2\eta - 1$ | $4\eta(1-\eta)$ |
| SVM | $\max(1-v,0)$ | $sign(2\eta - 1)$ | $1 - |2\eta - 1|$ |
| Boosting | $\exp(-v)$ | $\frac{1}{2}\log\frac{\eta}{1-\eta}$ | $2\sqrt{\eta(1-\eta)}$ |
| Logistic Regression | $\log(1+e^{-v})$ | $\log\frac{\eta}{1-\eta}$ | $-\eta\log\eta - (1-\eta)\log(1-\eta)$ |

(infinite) weight given to these points by the $\phi(\cdot)$ functions when $yf(\mathbf{x}) \to -\infty$. In this work, we show that this problem can be eliminated by allowing non-convex $\phi(\cdot)$. This may, at first thought, seem like a bad idea, given the widely held belief that the success of the aforementioned algorithms is precisely due to the convexity of these functions. We will see, however, that the convexity of $\phi(\cdot)$ is not important. What really matters is the fact, noted by [4], that the minimum conditional risk

$$C_\phi^*(\eta) = \inf_f C_\phi(\eta, f) = C_\phi(\eta, f_\phi^*) \qquad (6)$$

satisfies two properties. First, it is a concave function of $\eta$ ($\eta \in [0,1]$)[1]. Second, if $f_\phi^*$ is differentiable, then $C_\phi^*(\eta)$ is differentiable and, for any pair $(v, \hat{\eta})$ such that $v = f_\phi^*(\hat{\eta})$,

$$C_\phi(\eta, v) - C_\phi^*(\eta) = B_{-C_\phi^*}(\eta, \hat{\eta}), \qquad (7)$$

where

$$B_F(\eta, \hat{\eta}) = F(\eta) - F(\hat{\eta}) - (\eta - \hat{\eta})F'(\hat{\eta}). \qquad (8)$$

is the Bregman divergence of the convex function $F$. The second property provides an interesting interpretation of the learning algorithms as methods for the estimation of the class posterior probability $\eta(\mathbf{x})$: the search for the $f(\mathbf{x})$ which minimizes (4) is equivalent to a search for the probability estimate $\hat{\eta}(\mathbf{x})$ which minimizes (7). This raises the question of whether minimizing a cost of the form of (4) is the best way to elicit the posterior probability $\eta(\mathbf{x})$.

## 3 Probability elicitation

This question has been extensively studied in statistics. In particular, Savage studied the problem of designing reward functions that encourage probability forecasters to make accurate predictions [6]. The problem is formulated as follows.

- let $I_1(\hat{\eta})$ be the reward for the prediction $\hat{\eta}$ when the event $y = 1$ holds.
- let $I_{-1}(\hat{\eta})$ be the reward for the prediction $\hat{\eta}$ when the event $y = -1$ holds.

The expected reward is

$$I(\eta, \hat{\eta}) = \eta I_1(\hat{\eta}) + (1 - \eta)I_{-1}(\hat{\eta}). \qquad (9)$$

Savage asked the question of which functions $I_1(\cdot), I_{-1}(\cdot)$ make the expected reward maximal when $\hat{\eta} = \eta, \forall \eta$. These are the functions such that

$$I(\eta, \hat{\eta}) \le I(\eta, \eta) = J(\eta), \quad \forall \eta \qquad (10)$$

with equality if and only if $\hat{\eta} = \eta$. Using the linearity of $I(\eta, \hat{\eta})$ on $\eta$, and the fact that $J(\eta)$ is supported by $I(\eta, \hat{\eta})$ at, and only at, $\eta = \hat{\eta}$, this implies that $J(\eta)$ is strictly convex [6, 7]. Savage then showed that (10) holds if and only if

$$I_1(\eta) = J(\eta) + (1-\eta)J'(\eta) \qquad (11)$$
$$I_{-1}(\eta) = J(\eta) - \eta J'(\eta). \qquad (12)$$

Defining the loss of the prediction of $\eta$ by $\hat{\eta}$ as the difference to the maximum reward

$$L(\eta, \hat{\eta}) = I(\eta, \eta) - I(\eta, \hat{\eta})$$

Table 2: Probability elicitation form for various machine learning algorithms, and Savage's procedure. In Savage 1 and 2 $m' = m + k$.

| Algorithm | $I_1(\eta)$ | $I_{-1}(\eta)$ | $J(\eta)$ |
|---|---|---|---|
| Least squares | $-4(1-\eta)^2$ | $-4\eta^2$ | $-4\eta(1-\eta)$ |
| Modified LS | $-4(1-\eta)^2$ | $-4\eta^2$ | $-4\eta(1-\eta)$ |
| SVM | $sign[2\eta-1]-1$ | $sign[2\eta-1]+1$ | $|2\eta-1|-1$ |
| Boosting | $-\sqrt{\frac{1-\eta}{\eta}}$ | $-\sqrt{\frac{\eta}{1-\eta}}$ | $-2\sqrt{\eta(1-\eta)}$ |
| Log. Regression | $\log\eta$ | $\log(1-\eta)$ | $\eta\log\eta + (1-\eta)\log(1-\eta)$ |
| Savage 1 | $-k(1-\eta)^2 + m' + l$ | $-k\eta^2 + m$ | $k\eta^2 + l\eta + m$ |
| Savage 2 | $-k(1/\eta + \log\eta) + m' + l$ | $-k\log\eta + m'$ | $m + l\eta - k\log\eta$ |

it follows that

$$L(\eta, \hat{\eta}) = B_J(\eta, \hat{\eta}), \tag{13}$$

i.e. the loss is the Bregman divergence of $J$. Hence, for any probability $\eta$, the best prediction $\hat{\eta}$ is the one of minimum Bregman divergence with $\eta$. Savage went on to investigate which functions $J(\eta)$ are admissible. He showed that for losses of the form $L(\eta, \hat{\eta}) = H(h(\eta) - h(\hat{\eta}))$, with $H(0) = 0$ and $H(v) > 0, v \neq 0$, and $h(v)$ any function, only two cases are possible. In the first $h(v) = v$, i.e. the loss only depends on the difference $\eta - \hat{\eta}$, and the admissible $J$ are

$$J_1(\eta) = k\eta^2 + l\eta + m, \tag{14}$$

for some integers $(k, l, m)$. In the second $h(v) = \log(v)$, i.e. the loss only depends on the ratio $\eta/\hat{\eta}$, and the admissible $J$ are of the form

$$J_2(\eta) = m + l\eta - k\log\eta. \tag{15}$$

## 4   Classification vs. probability elicitation

The discussion above shows that the optimization carried out by the learning algorithms is identical to Savage's procedure for probability elicitation. Both procedures reduce to the search for

$$\hat{\eta}^* = \arg\min_{\hat{\eta}} B_F(\eta, \hat{\eta}), \tag{16}$$

where $F(\eta)$ is a convex function. In both cases, this is done indirectly. Savage starts from the specification of $F(\eta) = J(\eta)$, from which the conditional rewards $I_1(\eta)$ and $I_2(\eta)$ are derived, using (11) and (12). $\hat{\eta}^*$ is then found by maximizing the expected reward $I(\eta, \hat{\eta})$ of (9) with respect to $\hat{\eta}$. The learning algorithms start from the loss $\phi(\cdot)$. The conditional risk $C_\phi(\eta, f)$ is then minimized with respect to $f$, so as to obtain the minimum conditional risk $C_\phi^*(\eta)$ and the corresponding $f_\phi^*(\hat{\eta})$. This is identical to solving (16) with $F(\eta) = -C_\phi^*(\eta)$. Using the relation $J(\eta) = -C_\phi^*(\eta)$ it is possible to express the learning algorithms in "Savage form", i.e. as procedures for the maximization of (9), by deriving the conditional reward functions associated with each of the $C_\phi^*(\eta)$ in Table 1. This is done with (11) and (12) and the results are shown in Table 2. In all cases $I_1(\eta) = -\phi(f_\phi^*(\eta))$ and $I_{-1}(\eta) = -\phi(-f_\phi^*(\eta))$.

The opposite question of whether Savage's algorithms be expressed in "machine learning form", i.e. as the minimization of (4), is more difficult. It requires that the $I_i(\eta)$ satisfy

$$I_1(\eta) = -\phi(f(\eta)) \tag{17}$$
$$I_{-1}(\eta) = -\phi(-f(\eta)) \tag{18}$$

for some $f(\eta)$, and therefore constrains $J(\eta)$. To understand the relationship between $J$, $\phi$, and $f_\phi^*$ it helps to think of the latter as an inverse link function. Or, assuming that $f_\phi^*$ is invertible, to think of $\eta = (f_\phi^*)^{-1}(v)$ as a link function, which maps a real $v$ into a probability $\eta$. Under this interpretation, it is natural to consider link functions which exhibit the following symmetry

$$f^{-1}(-v) = 1 - f^{-1}(v). \tag{19}$$

Note that this implies that $f^{-1}(0) = 1/2$, i.e. $f$ maps $v = 0$ to $\eta = 1/2$. We refer to such link functions as symmetric, and show that they impose a special symmetry on $J(\eta)$.

Table 3: Probability elicitation form progresses from minimum conditional risk, and link function $(f_\phi^*)^{-1}(\eta)$, to loss $\phi$. $f_\phi^*(\eta)$ is not invertible for the SVM and modified LS methods.

| Algorithm | $J(\eta)$ | $(f_\phi^*)^{-1}(v)$ | $\phi(v)$ |
|---|---|---|---|
| Least squares | $-4\eta(1-\eta)$ | $\frac{1}{2}(v+1)$ | $(1-v)^2$ |
| Modified LS | $-4\eta(1-\eta)$ | NA | $\max(1-v,0)^2$ |
| SVM | $\|2\eta-1\|-1$ | N/A | $\max(1-v,0)$ |
| Boosting | $-2\sqrt{\eta(1-\eta)}$ | $\frac{e^{2v}}{1+e^{2v}}$ | $\exp(-v)$ |
| Logistic Regression | $\eta\log\eta+(1-\eta)\log(1-\eta)$ | $\frac{e^v}{1+e^v}$ | $\log(1+e^{-v})$ |

**Theorem 1.** *Let $I_1(\eta)$ and $I_{-1}(\eta)$ be two functions derived from a continuously differentiable function $J(\eta)$ according to (11) and (12), and $f(\eta)$ be an invertible function which satisfies (19). Then (17) and (18) hold if and only if*

$$J(\eta) = J(1-\eta). \tag{20}$$

*In this case,*

$$\phi(v) = -J[f^{-1}(v)] - (1 - f^{-1}(v))J'[f^{-1}(v)]. \tag{21}$$

The theorem shows that for any pair $J(\eta), f(\eta)$, such that $J(\eta)$ has the symmetry of (20) and $f(\eta)$ the symmetry of (19), the expected reward of (9) can be written in the "machine learning form" of (4), using (17) and (18) with the $\phi(v)$ given by (21). The following corollary specializes this result to the case where $J(\eta) = -C_\phi^*(\eta)$.

**Corollary 2.** *Let $I_1(\eta)$ and $I_{-1}(\eta)$ be two functions derived with (11) and (12) from any continuously differentiable $J(\eta) = -C_\phi^*(\eta)$, such that*

$$C_\phi^*(\eta) = C_\phi^*(1-\eta), \tag{22}$$

*and $f_\phi(\eta)$ be any invertible function which satisfies (19). Then*

$$I_1(\eta) = -\phi(f_\phi(\eta)) \tag{23}$$
$$I_{-1}(\eta) = -\phi(-f_\phi(\eta)) \tag{24}$$

*with*

$$\phi(v) = C_\phi^*[f_\phi^{-1}(v)] + (1 - f_\phi^{-1}(v))(C_\phi^*)'[f_\phi^{-1}(v)]. \tag{25}$$

Note that there could be many pairs $\phi, f_\phi$ for which the corollary holds[2]. Selecting a particular $f_\phi$ "pins down" $\phi$, according to (25). This is the case of the algorithms in Table 1, for which $C_\phi^*(\eta)$ and $f_\phi^*$ have the symmetries required by the corollary. The link functions associated with these algorithms are presented in Table 3. From these and (25) it is possible to recover $\phi(v)$, also shown in the table.

## 5 New loss functions

The discussion above provides an integrated picture of the "machine learning" and "probability elicitation" view of the classification problem. Table 1 summarizes the steps of the "machine learning view": start from the loss $\phi(v)$, and find 1) the inverse link function $f_\phi^*(\eta)$ of minimum conditional risk, and 2) the value of this risk $C_\phi^*(\eta)$. Table 3 summarizes the steps of the "probability elicitation view": start from 1) the expected maximum reward function $J(\eta)$ and 2) the link function $(f_\phi^*)^{-1}(v)$, and determine the loss function $\phi(v)$. If $J(\eta) = -C_\phi^*(\eta)$, the two procedures are equivalent, since they both reduce to the search for the probability estimate $\hat\eta^*$ of (16).

Comparing to Table 2, it is clear that the least squares procedures are special cases of Savage 1, with $k = -l = 4$ and $m = 0$, and the link function $\eta = (v+1)/2$. The constraint $k = -l$ is necessary

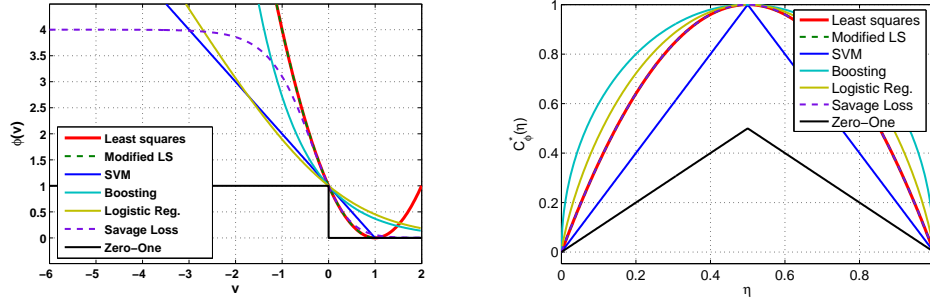

Figure 1: Loss function $\phi(v)$ (left) and minimum conditional risk $C_\phi^*(\eta)$ (right) associated with the different methods discussed in the text.

for (22) to hold, but not the others. For Savage 2, a "machine learning form" is not possible (at this point), because $J(\eta) \neq J(1-\eta)$. We currently do not know if such a form can be derived in cases like this, i.e. where the symmetries of (19) and/or (22) are absent. From the probability elicitation point of view, an important contribution of the machine learning research (in addition to the algorithms themselves) has been to identify new $J$ functions, namely those associated with the techniques other than least squares. From the machine learning point of view, the elicitation perspective is interesting because it enables the derivation of new $\phi$ functions.

The main observation is that, under the customary specification of $\phi$, both $C_\phi^*(\eta)$ and $f_\phi^*(\eta)$ are immediately set, leaving no open degrees of freedom. In fact, the selection of $\phi$ can be seen as the indirect selection of a link function $(f_\phi^*)^{-1}$ and a minimum conditional risk $C_\phi^*(\eta)$. The latter is an approximation to the minimum conditional risk of the *0-1 loss*, $C_{\phi_{0/1}}^*(\eta) = 1 - \max(\eta, 1 - \eta)$. The approximations associated with the existing algorithms are shown in Figure 1. The approximation error is smallest for the SVM, followed by least squares, logistic regression, and boosting, but all approximations are comparable. The alternative, suggested by the probability elicitation view, is to start with the selection of the approximation directly. In addition to allowing direct control over the quantity that is usually of interest (the minimum expected risk of the classifier), the selection of $C_\phi^*(\eta)$ (which is equivalent to the selection of $J(\eta)$) has the added advantage of leaving one degree of freedom open. As stated by Corollary 2 it is further possible to select across $\phi$ functions, by controlling the link function $f_\phi$. This allows tailoring properties of detail of the classifier, while maintaining its performance constant, in terms of the expected risk.

We demonstrate this point, by proposing a new loss function $\phi$. We start by selecting the minimum conditional risk of least squares (using Savage's version with $k = -l = 1, m = 0$) $C_\phi^*(\eta) = \eta(1-\eta)$, because it provides the best approximation to the Bayes error, while avoiding the lack of differentiability of the SVM. We next replace the traditional link function of least squares by the logistic link function (classically used with logistic regression) $f_\phi^* = \frac{1}{2} \log \frac{\eta}{1-\eta}$. When used in the context of boosting (LogitBoost [3]), this link function has been found less sensitive to outliers than other variants [8]. We then resort to (25) to find the $\phi$ function, which we denote by *Savage loss*,

$$\phi(v) = \frac{1}{(1 + e^{2v})^2}. \tag{26}$$

A plot of this function is presented in Figure 1, along with those associated with all the algorithms of Table 1. Note that the proposed loss is very similar to that of least squares in the region where $|v|$ is small (the margin), but quickly becomes constant as $v \to -\infty$. This is unlike all other previous $\phi$ functions, and suggests that classifiers designed with the new loss should be more robust to outliers.

It is also interesting to note that the new loss function is not convex, violating what has been an hallmark of the $\phi$ functions used in the literature. The convexity of $\phi$ is, however, not important, a fact that is made clear by the elicitation view. Note that the convexity of the expected reward of (9) only depends on the convexity of the functions $I_1(\eta)$ and $I_{-1}(\eta)$. These, in turn, only depend on the choice of $J(\eta)$, as shown by (11) and (12). From Corollary 2 it follows that, as long as the symmetries of (22) and (19) hold, and $\phi$ is selected according to (25), the selection of $C_\phi^*(\eta)$

**Algorithm 1 SavageBoost**

---

**Input:** Training set $\mathcal{D} = \{(\mathbf{x}_1, y_1), \ldots, (\mathbf{x}_n, y_n)\}$, where $y \in \{1, -1\}$ is the class label of example $\mathbf{x}$, and number $M$ of weak learners in the final decision rule.

**Initialization:** Select uniform weights $w_i^{(1)} = \frac{1}{|\mathcal{D}|}, \forall i$.

**for** $m = \{1, \ldots, M\}$ **do**

    compute the gradient step $G_m(\mathbf{x})$ with (30).

    update weights $w_i$ according to $w_i^{(m+1)} = w_i^{(m)} \times e^{y_i G_m(\mathbf{x}_i)}$.

**end for**

**Output:** decision rule $h(\mathbf{x}) = sgn[\sum_{m=1}^{M} G_m(\mathbf{x})]$.

---

completely determines the convexity of the conditional risk of (4). Whether $\phi$ is itself convex does not matter.

## 6 SavageBoost

We have hypothesized that classifiers designed with (26) should be more robust than those derived from the previous $\phi$ functions. To test this we designed a boosting algorithm based in the new loss, using the procedure proposed by Friedman to derive RealBoost [3]. At each iteration the algorithm searches for the weak learner $G(\mathbf{x})$ which further reduces the conditional risk $E_{Y|\mathbf{x}}[\phi(y(f(\mathbf{x}) + G(\mathbf{x})))|\mathbf{X} = \mathbf{x}]$ of the current $f(\mathbf{x})$, for every $\mathbf{x} \in \mathcal{X}$. The optimal weak learner is

$$G^*(\mathbf{x}) = \arg\min_{G(\mathbf{x})}\big\{\eta(\mathbf{x})\phi_w(G(\mathbf{x})) + (1 - \eta(\mathbf{x}))\phi_w(-G(\mathbf{x}))\big\} \tag{27}$$

where

$$\phi_w(yG(\mathbf{x})) = \frac{1}{(1 + w(\mathbf{x}, y)^2 e^{2y(G(\mathbf{x}))})^2} \tag{28}$$

and

$$w(\mathbf{x}, y) = e^{yf(\mathbf{x})} \tag{29}$$

The minimization is by gradient descent. Setting the gradient with respect to $G(\mathbf{x})$ to zero results in

$$G^*(\mathbf{x}) = \frac{1}{2}\left(\log \frac{P_w(y = 1|\mathbf{x})}{P_w(y = -1|\mathbf{x})}\right) \tag{30}$$

where $P_w(y = i|\mathbf{x})$ are probability estimates obtained from the re-weighted training set. At each iteration the optimal weak learner is found from (30) and reweighing is performed according to (29). We refer to the algorithm as *SavageBoost*, and summarize it in the inset.

## 7 Experimental results

We compared SavageBoost to AdaBoost [9], RealBoost [3], and LogitBoost [3]. The latter is generally considered more robust to outliers [8] and thus a good candidate for comparison. Ten binary UCI data sets were used: Pima-diabetes, breast cancer diagnostic, breast cancer prognostic, original Wisconsin breast cancer, liver disorder, sonar, echo-cardiogram, Cleveland heart disease, tic-tac-toe and Haberman's survival. We followed the training/testing procedure outlined in [2] to explore the robustness of the algorithms to outliers. In all cases, five fold validation was used with varying levels of outlier contamination. Figure 2 shows the average error of the four methods on the Liver-Disorder set. Table 4 shows the number of times each method produced the smallest error (#wins) over the ten data sets at a given contamination level, as well as the average error% over all data sets (at that contamination level). Our results confirm previous studies that have noted AdaBoost's sensitivity to outliers [1]. Among the previous methods AdaBoost indeed performed the worst, followed by RealBoost, with LogistBoost producing the best results. This confirms previous reports that LogitBoost is less sensitive to outliers [8]. SavageBoost produced generally better results than Ada and RealBoost at all contamination levels, including 0% contamination. LogitBoost achieves

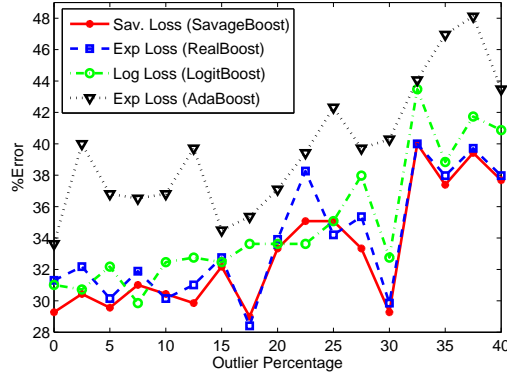

Figure 2: Average error for four boosting methods at different contamination levels.

Table 4: (number of wins, average error%) for each method and outlier percentage.

| Method | 0% outliers | 5% outliers | 40% outliers |
|---|---|---|---|
| Savage Loss (SavageBoost) | $(\mathbf{4}, \mathbf{19.22}\%)$ | $(\mathbf{4}, \mathbf{19.91}\%)$ | $(\mathbf{6}, \mathbf{25.9}\%)$ |
| Log Loss(LogitBoost) | $(4, 20.96\%)$ | $(4, 22.04\%)$ | $(3, 31.73\%)$ |
| Exp Loss(RealBoost) | $(2, 23.99\%)$ | $(2, 25.34\%)$ | $(0, 33.18\%)$ |
| Exp Loss(AdaBoost) | $(0, 24.58\%)$ | $(0, 26.45\%)$ | $(1, 38.22\%)$ |

comparable results at low contamination levels $(0\%, 5\%)$ but has higher error when contamination is significant. With $40\%$ contamination SavageBoost has 6 wins, compared to 3 for LogitBoost and, on average, about $6\%$ less error. Although, in all experiments, each algorithm was allowed 50 iterations, SavageBoost converged much faster than the others, requiring an average of 25 iterations at $0\%$ cantamination. This is in contrast to 50 iterations for LogitBoost and 45 iterations for RealBoost. We attribute fast convergence to the bounded nature of the new loss, that prevents so called "early stopping" problems [10]. Fast convergence is, of course, a great benefit in terms of the computational efficiency of training and testing. This issue will be studied in greater detail in the future.

## Footnotes

[1] Here, and throughout the paper, we omit the dependence of $\eta$ on $\mathbf{x}$, whenever we are referring to functions of $\eta$, i.e. mappings whose range is $[0,1]$.

[2]This makes the notation $f_\phi$ and $C_\phi^*$ technically inaccurate. $C_{f,\phi}^*$ would be more suitable. We, nevertheless, retain the $C_\phi^*$ notation for the sake of consistency with the literature.

# References

[1] T. G. Dietterich, "An experimental comparison of three methods for constructing ensembles of decision trees: Bagging, boosting, and randomization," *Machine Learning*, 2000.

[2] Y. Wu and Y. Liu, "Robust truncated-hinge-loss support vector machines," *JASA*, 2007.

[3] J. Friedman, T. Hastie, and R. Tibshirani, "Additive logistic regression: A statistical view of boosting," *Annals of Statistics*, 2000.

[4] T. Zhang, "Statistical behavior and consistency of classification methods based on convex risk minimization," *Annals of Statistics*, 2004.

[5] P. Bartlett, M. Jordan, and J. D. McAuliffe, "Convexity, classification, and risk bounds," *JASA*, 2006.

[6] L. J. Savage, "The elicitation of personal probabilities and expectations," *JASA*, vol. 66, pp. 783–801, 1971.

[7] S. Boyd and L. Vandenberghe, *Convex Optimization*. Cambridge: Cambridge University Press, 2004.

[8] R. McDonald, D. Hand, and I. Eckley, "An empirical comparison of three boosting algorithms on real data sets with artificial class noise," in *International Workshop on Multiple Classifier Systems*, 2003.

[9] Y. Freund and R. Schapire, "A decision-theoretic generalization of on-line learning and an application to boosting," *Journal of Computer and System Sciences*, 1997.

[10] T. Zhang and B. Yu, "Boosting with early stopping: Convergence and consistency," *Annals of Statistics*, 2005.

